# Inferring Neural Firing Rates from Spike Trains Using Gaussian Processes

**John P. Cunningham**[1], **Byron M. Yu**[1,2,3], **Krishna V. Shenoy**[1,2]
[1]Department of Electrical Engineering,
[2]Neurosciences Program, Stanford University, Stanford, CA 94305
{jcunnin,byronyu,shenoy}@stanford.edu

**Maneesh Sahani**[3]
[3]Gatsby Computational Neuroscience Unit, UCL
Alexandra House, 17 Queen Square, London, WC1N 3AR, UK
maneesh@gatsby.ucl.ac.uk

## Abstract

Neural spike trains present challenges to analytical efforts due to their noisy, spiking nature. Many studies of neuroscientific and neural prosthetic importance rely on a smoothed, denoised estimate of the spike train's underlying firing rate. Current techniques to find time-varying firing rates require *ad hoc* choices of parameters, offer no confidence intervals on their estimates, and can obscure potentially important single trial variability. We present a new method, based on a Gaussian Process prior, for inferring probabilistically optimal estimates of firing rate functions underlying single or multiple neural spike trains. We test the performance of the method on simulated data and experimentally gathered neural spike trains, and we demonstrate improvements over conventional estimators.

## 1 Introduction

Neuronal activity, particularly in cerebral cortex, is highly variable. Even when experimental conditions are repeated closely, the same neuron may produce quite different spike trains from trial to trial. This variability may be due to both randomness in the spiking process and to differences in cognitive processing on different experimental trials. One common view is that a spike train is generated from a smooth underlying function of time (the firing rate) and that this function carries a significant portion of the neural information. If this is the case, questions of neuroscientific and neural prosthetic importance may require an accurate estimate of the firing rate. Unfortunately, these estimates are complicated by the fact that spike data gives only a sparse observation of its underlying rate. Typically, researchers average across many trials to find a smooth estimate (averaging out spiking noise). However, averaging across many roughly similar trials can obscure important temporal features [1]. Thus, estimating the underlying rate from only one spike train (or a small number of spike trains believed to be generated from the same underlying rate) is an important but challenging problem.

The most common approach to the problem has been to collect spikes from multiple trials in a *peri-stimulus-time histogram* (PSTH), which is then sometimes smoothed by convolution or splines [2], [3]. Bin sizes and smoothness parameters are typically chosen *ad hoc* (but see [4], [5]) and the result is fundamentally a multi-trial analysis. An alternative is to convolve a single spike train with a kernel. Again, the kernel shape and time scale are frequently *ad hoc*. For multiple trials, researchers may average over multiple kernel-smoothed estimates. [2] gives a thorough review of classical methods.

More recently, point process likelihood methods have been adapted to spike data [6]–[8]. These methods optimize (implicitly or explicitly) the conditional intensity function $\lambda(t|x(t), H(t))$ — which gives the probability of a spike in $[t, t + dt)$, given an underlying rate function $x(t)$ and the history of previous spikes $H(t)$ — with respect to $x(t)$. In a regression setting, this rate $x(t)$ may be learned as a function of an observed covariate, such as a sensory stimulus or limb movement. In the unsupervised setting of interest here, it is constrained only by prior expectations such as smoothness. Probabilistic methods enjoy two advantages over kernel smoothing. First, they allow explicit modelling of interactions between spikes through the history term $H(t)$ (*e.g.*, refractory periods). Second, as we will see, the probabilistic framework provides a principled way to share information between trials and to select smoothing parameters.

In neuroscience, most applications of point process methods use maximum likelihood estimation. In the unsupervised setting, it has been most common to optimize $x(t)$ within the span of an arbitrary basis (such as a spline basis [3]). In other fields, a theory of generalized Cox processes has been developed, where the point process is conditionally Poisson, and $x(t)$ is obtained by applying a link function to a draw from a random process, often a Gaussian process (GP) (*e.g.* [9]). In this approach, parameters of the GP, which set the scale and smoothness of $x(t)$ can be learned by optimizing the (approximate) marginal likelihood or evidence, as in GP classification or regression. However, the link function, which ensures a nonnegative intensity, introduces possibly undesirable artifacts. For instance, an exponential link leads to a process that grows less smooth as the intensity increases.

Here, we make two advances. First, we adapt the theory of GP-driven point processes to incorporate a history-dependent conditional likelihood, suitable for spike trains. Second, we formulate the problem such that nonnegativity in $x(t)$ is achieved without a distorting link function or sacrifice of tractability. We also demonstrate the power of numerical techniques that makes application of GP methods to this problem computationally tractable. We show that GP methods employing evidence optimization outperform both kernel smoothing and maximum-likelihood point process models.

## 2   Gaussian Process Model For Spike Trains

Spike trains can often be well modelled by gamma-interval point processes [6], [10]. We assume the underlying nonnegative firing rate $x(t) : t \in [0, T]$ is a draw from a GP, and then we assume that our spike train is a conditionally inhomogeneous gamma-interval process (IGIP), given $x(t)$. The spike train is represented by a list of spike times $\mathbf{y} = \{y_0, \ldots, y_N\}$. Since we will model this spike train as an IGIP[1], $\mathbf{y} \mid x(t)$ is by definition a renewal process, so we can write:

$$p(\mathbf{y} \mid x(t)) = \prod_{i=1}^{N} p(y_i \mid y_{i-1}, x(t)) \cdot p_0(y_0 \mid x(t)) \cdot p_T(T \mid y_N, x(t)), \tag{1}$$

where $p_0(\cdot)$ is the density of the first spike occuring at $y_0$, and $p_T(\cdot)$ is the density of no spikes being observed on $(y_N, T]$; the density for IGIP intervals (of order $\gamma \geq 1$) (see *e.g.* [6]) can be written as:

$$p(y_i \mid y_{i-1}, x(t)) = \frac{\gamma x(y_i)}{\Gamma(\gamma)} \left( \gamma \int_{y_{i-1}}^{y_i} x(u) du \right)^{\gamma - 1} \exp\left\{ -\gamma \int_{y_{i-1}}^{y_i} x(u) du \right\}. \tag{2}$$

The true $p_0(\cdot)$ and $p_T(\cdot)$ under this gamma-interval spiking model are not closed form, so we simplify these distributions as intervals of an inhomogeneous Poisson process (IP). This step, which we find to sacrifice very little in terms of accuracy, helps to preserve tractability. Note also that we write the distribution in terms of the inter-spike-interval distribution $p(y_i|y_{i-1}, x(t))$ and not $\lambda(t|x(t), H(t))$, but the process could be considered equivalently in terms of conditional intensity.

We now discretize $x(t) : t \in [0, T]$ by the time resolution of the experiment ($\Delta$, here 1 ms), to yield a series of $n$ evenly spaced samples $\mathbf{x} = [x_1, \ldots, x_n]'$ (with $n = \frac{T}{\Delta}$). The events $\mathbf{y}$ become $N + 1$ time indices into $\mathbf{x}$, with $N$ much smaller than $n$. The discretized IGIP output process is now (ignoring terms that scale with $\Delta$):

$$p(\mathbf{y} \mid \mathbf{x}) = \prod_{i=1}^{N}\left[\frac{\gamma x_{y_i}}{\Gamma(\gamma)}\left(\gamma \sum_{k=y_{i-1}}^{y_i-1} x_k\Delta\right)^{\gamma-1}\exp\left\{-\gamma \sum_{k=y_{i-1}}^{y_i-1} x_k\Delta\right\}\right]$$

$$\cdot\, x_{y_0}\exp\left\{-\sum_{k=0}^{y_0-1} x_k\Delta\right\}\cdot\exp\left\{-\sum_{k=y_N}^{n-1} x_k\Delta\right\}, \tag{3}$$

where the final two terms are $p_0(\cdot)$ and $p_T(\cdot)$, respectively [11]. Our goal is to estimate a smoothly varying firing rate function from spike times. Loosely, instead of being restricted to only one family of functions, GP allows all functions to be possible; the choice of kernel determines which functions are more likely, and by how much. Here we use the standard squared exponential (SE) kernel. Thus, $\mathbf{x} \sim \mathcal{N}(\mu\mathbf{1}, \Sigma)$, where $\Sigma$ is the positive definite covariance matrix defined by

$$\Sigma = \left\{K(t_i, t_j)\right\}_{i,j\in\{1,\ldots,n\}} \text{ where } K(t_i, t_j) = \sigma_f^2\exp\left\{-\frac{\kappa}{2}(t_i - t_j)^2\right\} + \sigma_v^2\delta_{ij}. \tag{4}$$

For notational convenience, we define the hyperparameter set $\theta = [\mu; \gamma; \kappa; \sigma_f^2; \sigma_v^2]$. Typically, the GP mean $\mu$ is set to 0. Since our intensity function is nonnegative, however, it is sensible to treat $\mu$ instead as a hyperparameter and let it be optimized to a positive value. We note that other standard kernels - including the rational quadratic, Matern $\nu = \frac{3}{2}$, and Matern $\nu = \frac{5}{2}$ - performed similarly to the SE; thus we only present the SE here. For an in depth discussion of kernels and of GP, see [12].

As written, the model assumes only one observed spike train; it may be that we have $m$ trials believed to be generated from the same firing rate profile. Our method naturally incorporates this case: define $p(\{\mathbf{y}\}_1^m \mid \mathbf{x}) = \prod_{i=1}^{m} p(\mathbf{y}^{(i)} \mid \mathbf{x})$, where $\mathbf{y}^{(i)}$ denotes the $i$th spike train observed.[2] Otherwise, the model is unchanged.

## 3 Finding an Optimal Firing Rate Estimate

### 3.1 Algorithmic Approach

Ideally, we would calculate the posterior on firing rate $p(\mathbf{x} \mid \mathbf{y}) = \int_\theta p(\mathbf{x} \mid \mathbf{y}, \theta)p(\theta)d\theta$ (integrating over the hyperparameters $\theta$), but this problem is intractable. We consider two approximations: replacing the integral by evaluation at the modal $\theta$, and replacing the integral with a sum over a discrete grid of $\theta$ values. We first consider choosing a modal hyperparameter set (ML-II model selection, see [12]), i.e. $p(\mathbf{x} \mid \mathbf{y}) \approx q(\mathbf{x} \mid \mathbf{y}, \theta^*)$ where $q(\cdot)$ is some approximate posterior, and

$$\theta^* = \underset{\theta}{\operatorname{argmax}}\, p(\theta \mid \mathbf{y}) = \underset{\theta}{\operatorname{argmax}}\, p(\theta)p(\mathbf{y} \mid \theta) = \underset{\theta}{\operatorname{argmax}}\, p(\theta)\int_{\mathbf{x}} p(\mathbf{y} \mid \mathbf{x}, \theta)p(\mathbf{x} \mid \theta)d\mathbf{x}. \tag{5}$$

(This and the following equations hold similarly for a single observation $\mathbf{y}$ or multiple observations $\{\mathbf{y}\}_1^m$, so we consider only the single observation for notational brevity.) Specific choices for the hyperprior $p(\theta)$ are discussed in Results. The integral in Eq. 5 is intractable under the distributions we are modelling, and thus we must use an approximation technique. Laplace approximation and Expectation Propagation (EP) are the most widely used techniques (see [13] for a comparison). The Laplace approximation fits an unnormalized Gaussian distribution to the integrand in Eq. 5. Below we show this integrand is log concave in $\mathbf{x}$. This fact makes reasonable the Laplace approximation, since we know that the distribution being approximated is unimodal in $\mathbf{x}$ and shares log concavity with the normal distribution. Further, since we are modelling a non-zero mean GP, most of the Laplace approximated probability mass lies in the nonnegative orthant (as is the case with the true posterior). Accordingly, we write:

$$p(\mathbf{y} \mid \theta) = \int_{\mathbf{x}} p(\mathbf{y} \mid \mathbf{x}, \theta) p(\mathbf{x} \mid \theta) d\mathbf{x} \;\approx\; p(\mathbf{y} \mid \mathbf{x}^*, \theta) p(\mathbf{x}^* \mid \theta) \frac{(2\pi)^{\frac{n}{2}}}{|\Lambda^* + \Sigma^{-1}|^{\frac{1}{2}}}, \tag{6}$$

where $\mathbf{x}^*$ is the mode of the integrand and $\Lambda^* = -\nabla_{\mathbf{x}}^2 \log p(\mathbf{y} \mid \mathbf{x}, \theta) \mid_{\mathbf{x}=\mathbf{x}^*}$. Note that in general both $\Sigma$ and $\Lambda^*$ (and $\mathbf{x}^*$, implicitly) are functions of the hyperparameters $\theta$. Thus, Eq. 6 can be differentiated with respect to the hyperparameter set, and an iterative gradient optimization (we used conjugate gradients) can be used to find (locally) optimal hyperparameters. Algorithmic details and the gradient calculations are typical for GP; see [12]. The Laplace approximation also naturally provides confidence intervals from the approximated posterior covariance $(\Sigma^{-1} + \Lambda^*)^{-1}$.

We can also consider approximate integration over $\theta$ using the Laplace approximation above. The Laplace approximation produces a posterior approximation $q(\mathbf{x} \mid \mathbf{y}, \theta) = \mathcal{N}\left(\mathbf{x}^*, (\Lambda^* + \Sigma^{-1})^{-1}\right)$ and a model evidence approximation $q(\theta \mid \mathbf{y})$ (Eq. 6). The approximate integrated posterior can be written as $p(\mathbf{x} \mid \mathbf{y}) = E_{\theta \mid \mathbf{y}}[p(\mathbf{x} \mid \mathbf{y}, \theta)] \approx \sum_j q(\mathbf{x} \mid \mathbf{y}, \theta_j) q(\theta_j \mid \mathbf{y})$ for some choice of samples $\theta_j$ (which again gives confidence intervals on the estimates). Since the dimensionality of $\theta$ is small, and since we find in practice that the posterior on $\theta$ is well behaved (well peaked and unimodal), we find that a simple grid of $\theta_j$ works very well, thereby obviating MCMC or another sampling scheme. This approximate integration consistently yields better results than a modal hyperparameter set, so we will only consider approximate integration for the remainder of this report.

For the Laplace approximation at any value of $\theta$, we require the modal estimate of firing rate $\mathbf{x}^*$, which is simply the MAP estimator:

$$\mathbf{x}^* = \operatorname*{argmax}_{\mathbf{x} \succeq \mathbf{0}} p(\mathbf{x} \mid \mathbf{y}) = \operatorname*{argmax}_{\mathbf{x} \succeq \mathbf{0}} p(\mathbf{y} \mid \mathbf{x}) p(\mathbf{x}). \tag{7}$$

Solving this problem is equivalent to solving an unconstrained problem where $p(\mathbf{x})$ is a truncated multivariate normal (but this is not the same as individually truncating each marginal $p(x_i)$; see [14]). Typically a link or squashing function would be included to enforce nonnegativity in $\mathbf{x}$, but this can distort the intensity space in unintended ways. We instead impose the constraint $\mathbf{x} \succeq 0$, which reduces the problem to being solved over the (convex) nonnegative orthant. To pose the problem as a convex program, we define $f(\mathbf{x}) = -\log p(\mathbf{y} \mid \mathbf{x}) p(\mathbf{x})$:

$$f(\mathbf{x}) = \sum_{i=1}^{N} \left( -\log x_{y_i} - (\gamma - 1)\log \left( \sum_{k=y_{i-1}}^{y_i-1} x_k \Delta \right) \right) + \sum_{k=y_0}^{y_N-1} \gamma x_k \Delta \tag{8}$$

$$-\log x_{y_0} + \sum_{k=1}^{y_0-1} x_k \Delta + \sum_{k=y_N}^{n-1} x_k \Delta + \frac{1}{2}(\mathbf{x} - \mu\mathbf{1})^T \Sigma^{-1} (\mathbf{x} - \mu\mathbf{1}) + C, \tag{9}$$

where $C$ represents constants with respect to $\mathbf{x}$. From this form follows the Hessian

$$\nabla_{\mathbf{x}}^2 f(\mathbf{x}) = \Sigma^{-1} + \Lambda \text{ where } \Lambda = -\nabla_{\mathbf{x}}^2 \log p(\mathbf{y} \mid \mathbf{x}, \theta) = B + D, \tag{10}$$

where $D = \mathbf{diag}(x_{y_0}^{-2}, \ldots, 0, \ldots, x_{y_i}^{-2} \ldots, 0, \ldots, x_{y_N}^{-2})$ is positive semidefinite and diagonal. $B$ is block diagonal with $N$ blocks. Each block is rank 1 and associates its positive, nonzero eigenvalue with eigenvector $[0, \ldots, 0, \mathbf{b}_i^T, 0, \ldots, 0]^T$. The remaining $n - N$ eigenvalues are zero. Thus, $B$ has total rank $N$ and is positive semidefinite. Since $\Sigma$ is positive definite, it follows then that the Hessian is also positive definite, proving convexity. Accordingly, we can use a log barrier Newton method to efficiently solve for the global MAP estimator of firing rate $\mathbf{x}^*$ [15].

In the case of multiple spike train observations, we need only add extra terms of negative log likelihood from the observation model. This flows through to the Hessian, where $\nabla_{\mathbf{x}}^2 f(\mathbf{x}) = \Sigma^{-1} + \Lambda$ and $\Lambda = \Lambda_1 + \ldots + \Lambda_m$, with $\Lambda_i \; \forall i \in \{1, \ldots, m\}$ defined for each observation as in Eq. 10.

## 3.2 Computational Practicality

This method involves multiple iterative layers which require many Hessian inversions and other matrix operations (matrix-matrix products and determinants) that cost $\mathcal{O}(n^3)$ in run-time complexity and $\mathcal{O}(n^2)$ in memory, where $(\mathbf{x} \in \mathbb{R}^n)$. For any significant data size, a straightforward implementation is hopelessly slow. With 1 ms time resolution (or similar), this method would be restricted to spike trains lasting less than a second, and even this problem would be burdensome. Achieving computational improvements is critical, as a naive implementation is, for all practical purposes, intractable. Techniques to improve computational performance are a subject of study in themselves and are beyond the scope of this paper. We give a brief outline in the following paragraph.

In the MAP estimation of $\mathbf{x}^*$, since we have analytical forms of all matrices, we avoid explicit representation of any matrix, resulting in linear storage. Hessian inversions are avoided using the matrix inversion lemma and conjugate gradients, leaving matrix vector multiplications as the single costly operation. Multiplication of any vector by $\Lambda$ can be done in linear time, since $\Lambda$ is a (blockwise) vector outer product matrix. Since we have evenly spaced resolution of our data $\mathbf{x}$ in time indices $t_i$, $\Sigma$ is Toeplitz; thus multiplication by $\Sigma$ can be done using Fast Fourier Transform (FFT) methods [16]. These techniques allow exact MAP estimation with linear storage and nearly linear run time performance. In practice, for example, this translates to solving MAP estimation problems of $10^3$ variables in fractions of a second, with minimal memory load. For the modal hyperparameter scheme (as opposed to approximately integrating over the hyperparameters), gradients of Eq. 6 must also be calculated at each step of the model evidence optimization. In addition to using similar techniques as in the MAP estimation, log determinants and their derivatives (associated with the Laplace approximation) can be accurately approximated by exploiting the eigenstructure of $\Lambda$.

In total, these techniques allow optimal firing rates functions of $10^3$ to $10^4$ variables to be estimated in seconds or minutes (on a modern workstation). These data sizes translate to seconds of spike data at 1 ms resolution, long enough for most electrophysiological trials. This algorithm achieves a reduction from a naive implementation which would require large amounts of memory and would require many hours or days to complete.

## 4 Results

We tested the methods developed here using both simulated neural data, where the true firing rate was known by construction, and in real neural spike trains, where the true firing rate was estimated by a PSTH that averaged many similar trials. The real data used were recorded from macaque premotor cortex during a reaching task (see [17] for experimental method). Roughly 200 repeated trials per neuron were available for the data shown here.

We compared the IGIP-likelihood GP method (hereafter, GP IGIP) to other rate estimators (kernel smoothers, Bayesian Adaptive Regressions Splines or BARS [3], and variants of the GP method) using root mean squared difference (RMS) to the true firing rate. PSTH and kernel methods approximate the mean conditional intensity $\overline{\lambda}(t) = E_{H(t)}[\lambda(t|x(t), H(t))]$. For a renewal process, we know (by the time rescaling theorem [7], [11]) that $\overline{\lambda}(t) = x(t)$, and thus we can compare the GP IGIP (which finds $x(t)$) directly to the kernel methods. To confirm that hyperparameter optimization improves performance, we also compared GP IGIP results to maximum likelihood (ML) estimates of $x(t)$ using fixed hyperparameters $\theta$. This result is similar in spirit to previously published likelihood methods with fixed bases or smoothness parameters. To evaluate the importance of an observation model with spike history dependence (the IGIP of Eq. 3), we also compared GP IGIP to an inhomogeneous Poisson (GP IP) observation model (again with a GP prior on $x(t)$; simply $\gamma = 1$ in Eq. 3).

The hyperparameters $\theta$ have prior distributions ($p(\theta)$ in Eq. 5). For $\sigma_f$, $\kappa$, and $\gamma$, we set lognormal priors to enforce meaningful values (*i.e.* finite, positive, and greater than 1 in the case of $\gamma$). Specifically, we set $\log(\sigma_f^2) \sim \mathcal{N}(5, 2)$, $\log(\kappa) \sim \mathcal{N}(2, 2)$, and $\log(\gamma - 1) \sim \mathcal{N}(0, 100)$. The variance $\sigma_v$ can be set arbitrarily small, since the GP IGIP method avoids explicit inversions of $\Sigma$ with the matrix inversion lemma (see 3.2). For the approximate integration, we chose a grid consisting of the empirical mean rate for $\mu$ (that is, total spike count $N$ divided by total time $T$) and $(\gamma, \log(\sigma_f^2), \log(\kappa)) \in [1, 2, 4] \times [4, \ldots, 8] \times [0, \ldots, 7]$. We found this coarse grid (or similar) produced similar results to many other very finely sampled grids.

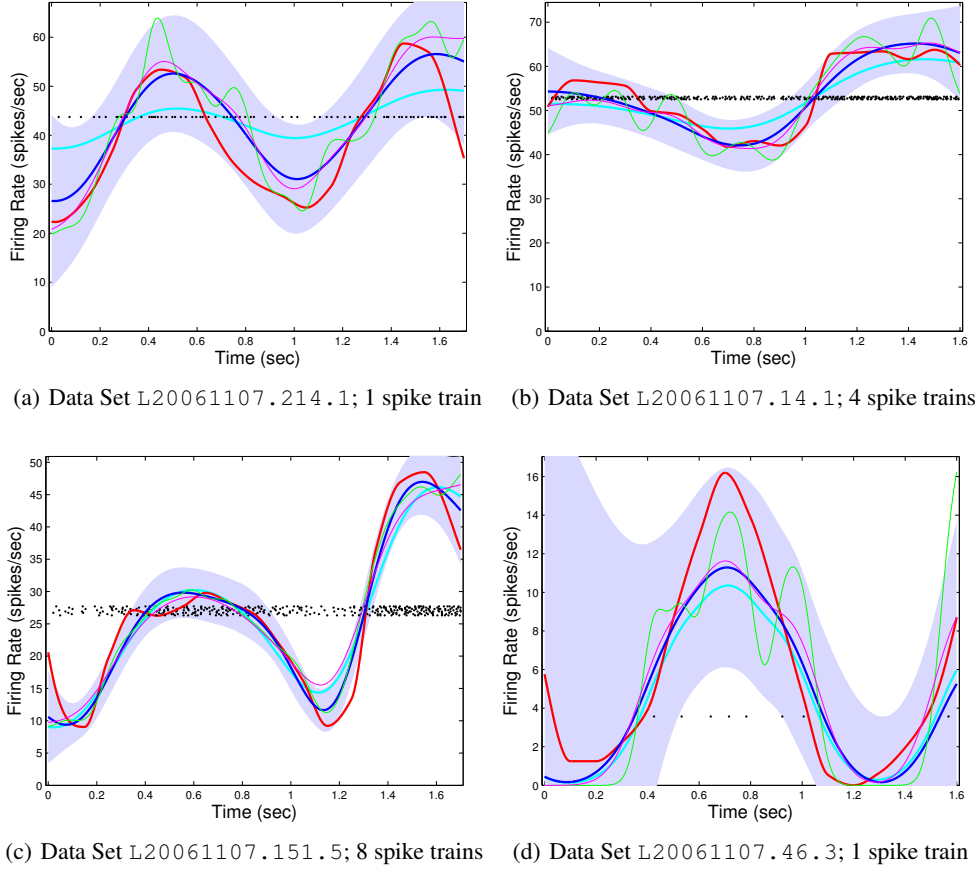

(a) Data Set `L20061107.214.1`; 1 spike train    (b) Data Set `L20061107.14.1`; 4 spike trains

(c) Data Set `L20061107.151.5`; 8 spike trains    (d) Data Set `L20061107.46.3`; 1 spike train

Figure 1: Sample GP firing rate estimate. See full description in text.

The four examples in Fig. 1 represent experimentally gathered firing rate profiles (according to the methods in [17]). In each of the plots, the empirical average firing rate of the spike trains is shown in bold red. For simulated spike trains, the spike trains were generated from each of these empirical average firing rates using an IGIP ($\gamma = 4$, comparable to fits to real neural data). For real neural data, the spike train(s) were selected as a subset of the roughly 200 experimentally recorded spike trains that were used to construct the firing rate profile. These spike trains are shown as a train of black dots, each dot indicating a spike event time (the y-axis position is not meaningful). This spike train or group of spike trains is the only input given to each of the fitting models. In thin green and magenta, we have two kernel smoothed estimates of firing rates; each represents the spike trains convolved with a normal distribution of a specified standard deviation (50 and 100 ms). We also smoothed these spike trains with adaptive kernel [18], fixed ML (as described above), BARS [3], and 150 ms kernel smoothers. We do not show these latter results in Fig. 1 for clarity of figures. These standard methods serve as a baseline from which we compare our method. In bold blue, we see $\mathbf{x}^*$, the results of the GP IGIP method. The light blue envelopes around the bold blue GP firing rate estimate represent the 95% confidence intervals. Bold cyan shows the GP IP method. This color scheme holds for all of Fig. 1.

We then ran all methods 100 times on each firing rate profile, using (separately) simulated and real neural spike trains. We are interested in the average performance of GP IGIP vs. other GP methods (a fixed ML or a GP IP) and vs. kernel smoothing and spline (BARS) methods. We show these results in Fig. 2. The four panels correspond to the same rate profiles shown in Fig. 1. In each panel, the top, middle, and bottom bar graphs correspond to the method on 1, 4, and 8 spike trains, respectively. GP IGIP produces an average RMS error, which is an improvement (or, less often, a deterioration) over a competing method. Fig. 2 shows the percent improvement of the GP IGIP method vs. the competing method listed. Only significant results are shown (paired t-test, $p < 0.05$).

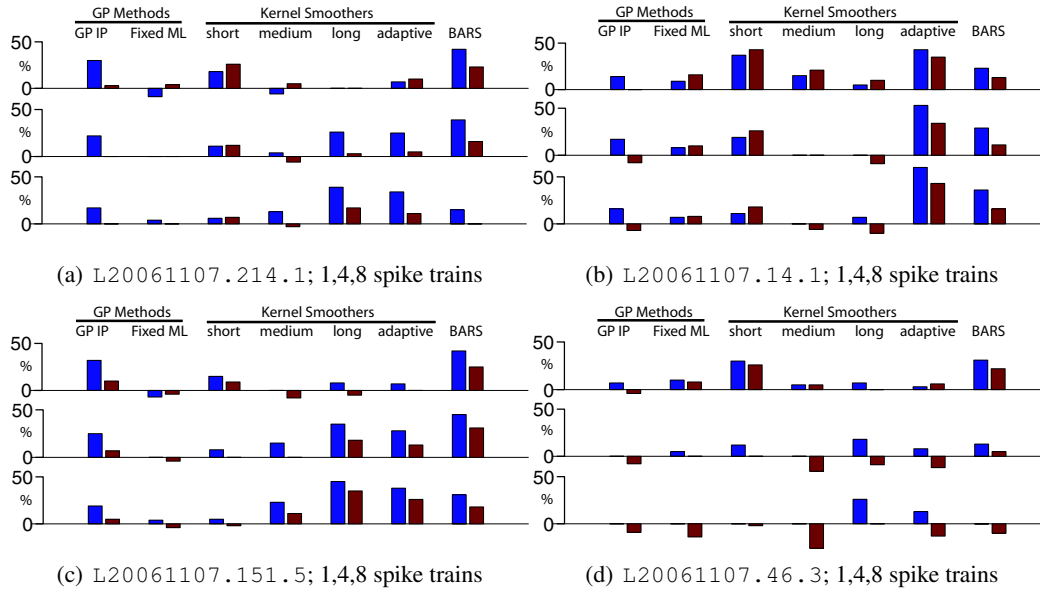

(a) `L20061107.214.1`; 1,4,8 spike trains

(b) `L20061107.14.1`; 1,4,8 spike trains

(c) `L20061107.151.5`; 1,4,8 spike trains

(d) `L20061107.46.3`; 1,4,8 spike trains

Figure 2: Average percent RMS improvement of GP IGIP method (with model selection) vs. method indicated in the column title. See full description in text.

Blue improvement bars are for simulated spike trains; red improvement bars are for real neural spike trains. The general positive trend indicates improvements, suggesting the utility of this approach. Note that, in the few cases where a kernel smoother performs better (*e.g.* the long bandwidth kernel in panel (b), real spike trains, 4 and 8 spike trains), outperforming the GP IGIP method requires an optimal kernel choice, which can not be judged from the data alone. In particular, the adaptive kernel method generally performed more poorly than GP IGIP. The relatively poor performance of GP IGIP vs. different techniques in panel (d) is considered in the Discussion section. The data sets here are by no means exhaustive, but they indicate how this method performs under different conditions.

## 5   Discussion

We have demonstrated a new method that accurately estimates underlying neural firing rate functions and provides confidence intervals, given one or a few spike trains as input. This approach is not without complication, as the technical complexity and computational effort require special care. Estimating underlying firing rates is especially challenging due to the inherent noise in spike trains. Having only a few spike trains deprives the method of many trials to reduce spiking noise. It is important here to remember why we care about single trial or small number of trial estimates, since we believe that in general the neural processing on repeated trials is not identical. Thus, we expect this signal to be difficult to find with or without trial averaging.

In this study we show both simulated and real neural spike trains. Simulated data provides a good test environment for this method, since the underlying firing rate is known, but it lacks the experimental proof of real neural spike trains (where spiking does not exactly follow a gamma-interval process). For the real neural spike trains, however, we do not know the true underlying firing rate, and thus we can only make comparisons to a noisy, trial-averaged mean rate, which may or may not accurately reflect the true underlying rate of an individual spike train (due to different cognitive processing on different trials). Taken together, however, we believe the real and simulated data give good evidence of the general improvements offered by this method.

Panels (a), (b), and (c) in Fig. 2 show that GP IGIP offers meaningful improvements in many cases and a small loss in performance in a few cases. Panel (d) tells a different story. In simulation, GP IGIP generally outperforms the other smoothers (though, by considerably less than in other panels). In real neural data, however, GP IGIP performs the same or relatively worse than other methods. This may indicate that, in the low firing rate regime, the IGIP is a poor model for real neural spiking.

It may also be due to our algorithmic approximations (namely, the Laplace approximation, which allows density outside the nonnegative orthant). We will report on this question in future work.

Furthermore, some neural spike trains may be inherently ill-suited to analysis. A problem with this and any other method is that of very low firing rates, as only occasional insight is given into the underlying generative process. With spike trains of only a few spikes/sec, it will be impossible for any method to find interesting structure in the firing rate. In these cases, only with many trial averaging can this structure be seen.

Several studies have investigated the inhomogeneous gamma and other more general models (*e.g.* [6], [19]), including the inhomogeneous inverse gaussian (IIG) interval and inhomogeneous Markov interval (IMI) processes. The methods of this paper apply immediately to any log-concave inhomogeneous renewal process in which inhomogeneity is generated by time-rescaling (this includes the IIG and several others). The IMI (and other more sophisticated models) will require some changes in implementation details; one possibility is a variational Bayes approach. Another direction for this work is to consider significant nonstationarity in the spike data. The SE kernel is standard, but it is also stationary; the method will have to compromise between areas of categorically different covariance. Nonstationary covariance is an important question in modelling and remains an area of research [20]. Advances in that field should inform this method as well.

## Acknowledgments

This work was supported by NIH-NINDS-CRCNS-R01, the Michael Flynn SGF, NSF, NDSEGF, Gatsby, CDRF, BWF, ONR, Sloan, and Whitaker. This work was conceived at the UK Spike Train Workshop, Newcastle, UK, 2006; we thank Stuart Baker for helpful discussions during that time. We thank Vikash Gilja, Stephen Ryu, and Mackenzie Risch for experimental, surgical, and animal care assistance. We thank also Araceli Navarro.

## Footnotes

[1]The IGIP is one of a class of renewal models that works well for spike data (much better than inhomogeneous Poisson; see [6], [10]). Other log-concave renewal models such as the inhomogeneous inverse-Gaussian interval can be chosen, and the implementation details remain unchanged.

[2]Another reasonable approach would consider each trial as having a different rate function $\mathbf{x}$ that is a draw from a GP with a nonstationary mean function $\boldsymbol{\mu}(t)$. Instead of inferring a mean rate function $\mathbf{x}^*$, we would learn a distribution of means. We are considering this choice for future work.

## References

[1] B. Yu, A. Afshar, G. Santhanam, S. Ryu, K. Shenoy, and M. Sahani. *Advances in NIPS*, 17, 2005.

[2] R. Kass, V. Ventura, and E. Brown. *J. Neurophysiol*, 94:8–25, 2005.

[3] I. DiMatteo, C. Genovese, and R. Kass. *Biometrika*, 88:1055–1071, 2001.

[4] H. Shimazaki and S. Shinomoto. *Neural Computation*, 19(6):1503–1527, 2007.

[5] D. Endres, M. Oram, J. Schindelin, and P. Foldiak. *Advances in NIPS*, 20, 2008.

[6] R. Barbieri, M. Quirk, L. Frank, M. Wilson, and E. Brown. *J Neurosci Methods*, 105:25–37, 2001.

[7] E. Brown, R. Barbieri, V. Ventura, R. Kass, and L. Frank. *Neural Comp*, 2002.

[8] W. Truccolo, U. Eden, M. Fellows, J. Donoghue, and E. Brown. *J Neurophysiol.*, 93:1074–1089, 2004.

[9] J. Moller, A. Syversveen, and R. Waagepetersen. *Scandanavian J. of Stats.*, 1998.

[10] K. Miura, Y. Tsubo, M. Okada, and T. Fukai. *J Neurosci.*, 27:13802–13812, 2007.

[11] D. Daley and D. Vere-Jones. *An Introduction to the Theory of Point Processes*. Springer, 2002.

[12] C. Rasmussen and C. Williams. *Gaussian Processes for Machine Learning*. MIT Press, 2006.

[13] M. Kuss and C. Rasmussen. *Journal of Machine Learning Res.*, 6:1679–1704, 2005.

[14] W. Horrace. *J Multivariate Analysis*, 94(1):209–221, 2005.

[15] S. Boyd and L. Vandenberghe. *Convex Optimization*. Cambridge University Press, 2004.

[16] B. Silverman. *Journal of Royal Stat. Soc. Series C: Applied Stat.*, 33, 1982.

[17] C. Chestek, A. Batista, G. Santhanam, B. Yu, A. Afshar, J. Cunningham, V. Gilja, S. Ryu, M. Churchland, and K. Shenoy. *J Neurosci.*, 27:10742–10750, 2007.

[18] B. Richmond, L. Optican, and H. Spitzer. *J. Neurophys.*, 64(2), 1990.

[19] R. Kass and V. Ventura. *Neural Comp*, 14:5–15, 2003.

[20] C. Paciorek and M. Schervish. *Advances in NIPS*, 15, 2003.

